# Boosting Decision Trees

**Harris Drucker**
AT&T Bell Laboratories
Holmdel, New Jersey 07733

**Corinna Cortes**
AT&T Bell Laboratories
Murray Hill, New Jersey 07974

## Abstract

A new boosting algorithm of Freund and Schapire is used to improve the performance of decision trees which are constructed using the information ratio criterion of Quinlan's C4.5 algorithm. This boosting algorithm iteratively constructs a series of decision trees, each decision tree being trained and pruned on examples that have been filtered by previously trained trees. Examples that have been incorrectly classified by the previous trees in the ensemble are resampled with higher probability to give a new probability distribution for the next tree in the ensemble to train on. Results from optical character recognition (OCR), and knowledge discovery and data mining problems show that in comparison to single trees, or to trees trained independently, or to trees trained on subsets of the feature space, the boosting ensemble is much better.

## 1 INTRODUCTION

A new boosting algorithm termed **AdaBoost** by their inventors (Freund and Schapire, 1995) has advantages over the original boosting algorithm (Schapire, 1990) and a second version (Freund, 1990). The implications of a boosting algorithm is that one can take a series of learning machines (termed weak learners) each having a poor error rate (but no worse than $.5-\gamma$, where $\gamma$ is some small positive number) and combine them to give an ensemble that has very good performance (termed a strong learner). The first practical implementation of boosting was in OCR (Drucker, 1993, 1994) using neural networks as the weak learners. In a series of comparisons (Bottou, 1994) boosting was shown to be superior to other techniques on a large OCR problem.

The general configuration of **AdaBoost** is shown in Figure 1. Each box is a decision tree built using Quinlans C4.5 algorithm (Quinlan, 1993) The key idea is that each weak learner is trained sequentially. The first weak learner is trained on a set of patterns picked randomly (with replacement) from a training set. After training and pruning, the training patterns are passed through this first decision tree. In the two class case the hypothesis $h_1$ is either class 0 or class 1. Some of the patterns will be in error. The training set for the

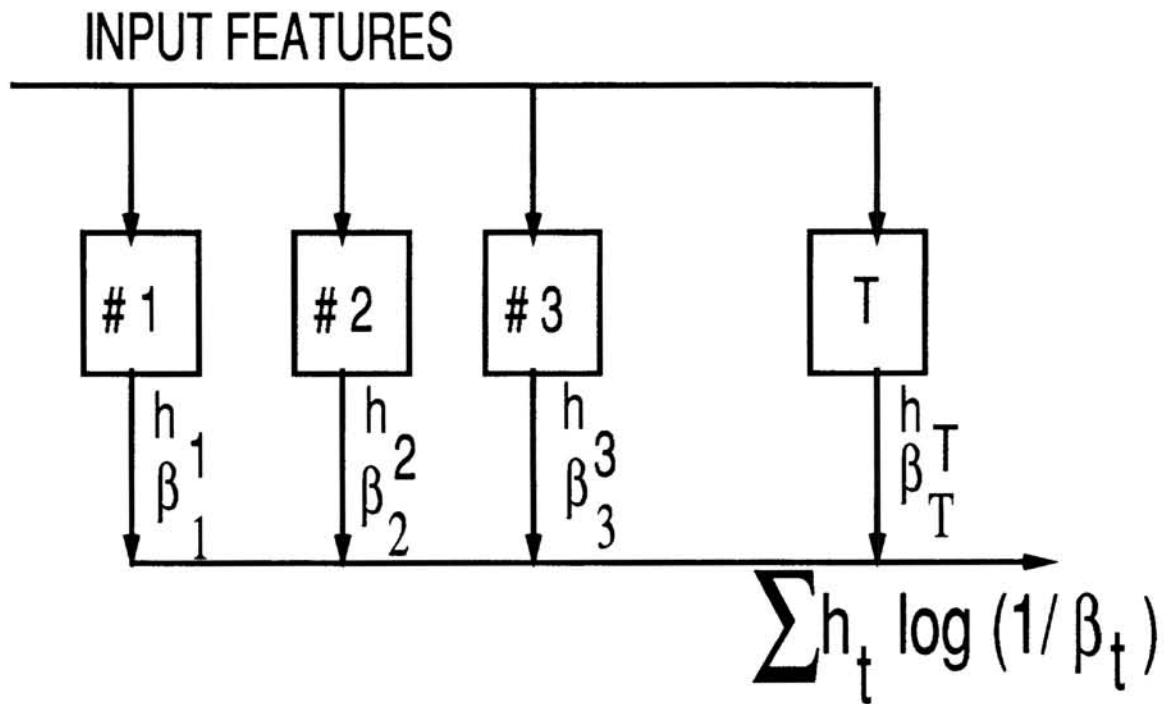

FIGURE 1. BOOSTING ENSEMBLE

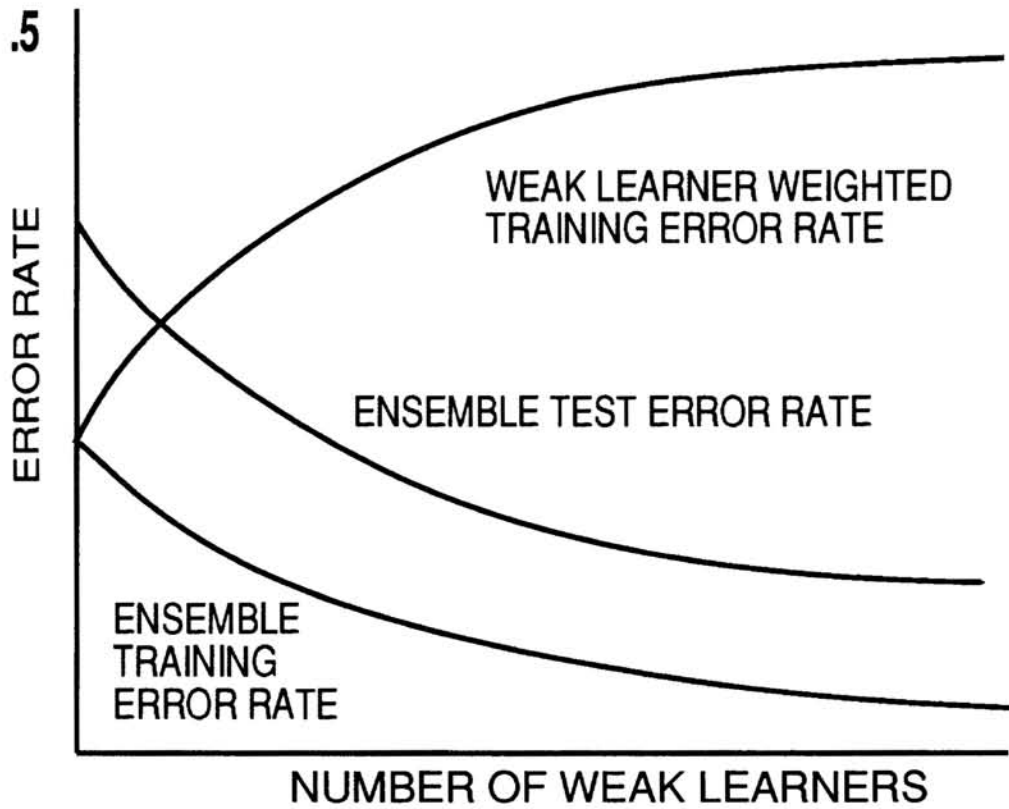

FIGURE 2. INDIVIDUAL WEAK LEARNER ERROR RATE
AND ENSEMBLE TRAINING AND TEST ERROR RATES

second weak learner will consist of patterns picked from the training set with higher probability assigned to those patterns the first weak learner classifies incorrectly. Since patterns are picked with replacement, difficult patterns are more likely to occur multiple times in the training set. Thus as we proceed to build each member of the ensemble, patterns which are more difficult to classify correctly appear more and more likely. The training error rate of an *individual* weak learner tends to grow as we increase the number of weak learners because each weak learner is asked to classify progressively more difficult patterns. However the boosting algorithm shows us that the *ensemble* training and test error rate decrease as we increase the number of weak learners. The ensemble output is determined by weighting the hypotheses with the log of $(1/\beta_i)$ where $\beta$ is proportional to the weak learner error rate. If the weak learner has good error rate performance, it will contribute significantly to the output, because then $1/\beta$ will be large.

Figure 2 shows the general shape of the curves we would expect. Say we have constructed N weak learners where N is a large number (right hand side of the graph). The N'th weak learner (top curve) will have a training error rate that approaches .5 because it is trained on difficult patterns and can do only sightly better than guessing. The bottom two curves show the test and training error rates of the ensemble using **all** N weak learners. which decrease as weak learners are added to the ensemble.

## 2 BOOSTING

Boosting arises from the PAC (probably approximately correct) learning model which has as one of its primary interests the efficiency of learning. Schapire was the first one to show that a series of weak learners could be converted to a strong learner. The detailed algorithm is show in Figure 3. Let us call the set of $N_1$ distinct examples the **original** training set. We distinguish the original training set from what we will call the **filtered** training set which consists of $N_1$ examples picked with replacement from the original training set. Basically each of $N_1$ original examples is assigned a weight which is proportional to the probability that the example will appear in the filtered training set (these weights have nothing to do with the weights usually associated with neural networks). Initially all examples are assigned a weight of unity so that all the examples are equally likely to show up in the initial set of training examples. However, the weights are altered at each state of boosting (Step 5 of Figure 3) and if the weights are high we may have multiple copies of some of the original examples appearing in the filtered training set. In step three of this algorithm, we calculate what is called the weighted training error and this is the error rate over all the **original** $N_1$ training examples weighted by their current respective probabilities. The algorithms terminates if this error rate is .5 (no better than guessing) or zero (then the weights of step 5 do not change). Although not called for in the original C4.5 algorithm, we also have an original set of pruning examples which also are assigned weights to form a filtered pruning set and used to prune the classification trees constructed using the filtered training set. It is known (Mingers, 1989a) that reducing the size of the tree (pruning) improves generalization.

## 3 DECISION TREES

For our implementation of decision trees, we have a set of features (attributes) that specifies an example along with their classification (we discuss the two-class problem primarily). We pick a feature that based on some criterion, best splits the examples into two subsets. Each of these two subsets will usually not contain examples of just one class, so we recursively divide the subsets until the final subsets each contain examples of just one class. Thus, each internal node specifies a feature and a value for that feature that determines whether one should take the left or right branch emanating from that node. At terminal nodes, we make the final decision, class 0 or 1. Thus, in decision trees one starts at a root node and progressively traverses the tree from the root node to one of the

Inputs:  $N_1$ training patterns, $N_2$ pruning patterns, $N_3$ test patterns

**Initialize** the weight vector of the $N_1$ training patterns: $w_i^1 = 1$ for $i=1,...,N_1$
**Initialize** the weight vector of the $N_2$ pruning patterns: $s_i^1 = 1$ for $i=1,...,N_2$
**Initialize** the number of trees in the ensemble to $t = 1$

**Do Until** weighted training error rate is 0 or .5 or ensemble test error rate asymptotes

1. For the training set and pruning sets

$$p^t = \frac{w^t}{\sum\limits_{i=1}^{N_1} w_i^t} \qquad\qquad r^t = \frac{s^t}{\sum\limits_{i=1}^{N_2} s_i^t}$$

Pick $N_1$ samples from original training set with probability p(i) to form filtered training set.
Pick $N_2$ samples from original pruning set with probability r(i) to form filtered pruning set.

2. Train tree t using filtered training set and prune using filtered pruning set

3. Pass the $N_1$ original training examples through the pruned tree whose output $h_t(i)$ is either 0 or 1 and classification c(i) is either 0 or 1. Calculate the weighted training error
rate: $\varepsilon_t = \sum\limits_{i=1}^{N_1} p_i^t |h_t(i) - c(i)|$

4. Set $\beta_t = \dfrac{\varepsilon_t}{1 - \varepsilon_t}$

5. Set the new training weight vector to be

$$w_i^{t+1} = w_i^t \{\beta_t^{**}(1-|h_t(i) - c(i)|)\} \qquad i = 1,...,N_1$$

Pass the $N_2$ original pruning patterns through the pruned tree and calculate new pruning weight vector:

$$s_i^{t+1} = s_i^t \{\beta_t^{**}(1-|h_t(i) - c(i)|)\} \qquad i = 1,...,N_2$$

6. For each tree t in the ensemble (total trees T) , pass the j'th test pattern through and obtain $h_t(j)$ for each t. The final hypothesis $h_f(j)$ for this pattern:

$$h_f(j) = \begin{cases} 1, & \sum\limits_{t=1}^{T}(\log\frac{1}{\beta_t})\, h_t(j) \geq \frac{1}{2}\sum\limits_{t=1}^{T}\log\frac{1}{\beta_t} \\ 0, & \text{otherwise} \end{cases}$$

Do for each test pattern and calculate the ensemble test error rate:

7. $t = t + 1$

**End_Until**

Figure 3: Boosting Algorithm

terminal nodes where a final decision is made. CART (Breiman, 1984) and C4.5 (Quinlan 1993) are perhaps the two most popular tree building algorithms. Here, C4.5 is used. The attraction of trees is that the simplest decision tree can be respecified as a series of rules and for certain potential users this is more appealing than a nonlinear "black box" such as a neural network. That is not to say that one can not design trees where the decision at each node depends on some nonlinear combination of features, but this will not be our implementation.

Other attractions of decision trees are speed of learning and evaluation. Whether trees are more accurate than other techniques depends on the application domain and the effectiveness of the particular implementation. In OCR, our neural networks are more accurate than trees but the penalty is in training and evaluation times. In other applications which we will discuss later a boosting network of trees is more accurate. As an initial example of the power of boosting, we will use trees for OCR of hand written digits. The main rationale for using OCR applications to evaluate **AdaBoost** is that we have experience in the use of a competing technology (neural networks) and we have from the National Institute of Standards and Technology (NIST) a large database of 120,000 digits, large enough so we can run multiple experiments. However, we will not claim that trees for OCR have the best error performance.

Once the tree is constructed, it is pruned to give hopefully better generalization performance than if the original tree was used. C4.5 uses the original training set for what is called "pessimistic pruning" justified by the fact that there may not be enough extra examples to form a set of pruning examples. However, we prefer to use an independent set of examples to prune this tree. In our case, we have (for each tree in the ensemble) an independent filtered pruning set of examples whose statistical distribution is similar to that of the filtered training set. Since the filtering imposed by the previous members of the ensemble can severely distort the original training distribution, we trust this technique more than pessimistic pruning. In pruning (Mingers, 1989), we pass the pruning set though the tree recording at each node (including non-terminal nodes) how many errors there would be if the tree was terminated there. Then, for each node (except for terminal nodes), we examine the subtree of that node. We then calculate the number of errors that would be obtained if that node would be made a terminal node and compare it to the number of errors at the terminal nodes of that subtree. If the number of errors at the root node of this subtree is less than or equal to that of the subtree, we replace the subtree with that node and make it a terminal node. Pruning tends to substantially reduce the size of the tree, even if the error rates are not substantially decreased.

## 4 EXPERIMENTS

In order to run enough experiments to claim statistical validity we needed a large supply of data and few enough features that the information ratio could be determined in a reasonable amount of time. Thus we used the 120,000 examples in a NIST database of digits subsampled to give us a 10x10 pixel array (100 features) where the features are continuous values. We do not claim that OCR is best done by using classification trees and certainly not in 100-dimensional space. We used 10,000 training examples, 2000 pruning examples and 2000 test examples for a total of 14,000 examples.

We also wanted to test our techniques on a wide range of problems, from easy to hard. Therefore, to make the problem reasonably difficult, we assigned class 0 to all digits from 0 to 4 (inclusive) and assigned class 1 to the remainder of the digits. To vary the difficulty of the problem, we prefiltered the data to form data sets of difficulty $f$. Think of $f$ as the fraction of hard examples generated by passing the 120,000 examples through a poorly trained neural network and accepting the misclassified examples with probability $f$ and the correctly classified examples with probability $1 - f$. Thus $f = .9$ means that the training set consists of 10,000 examples that if passed through this neural network would

have an error rate of .9. Table I compares the boosting performance with single tree performance. Also indicated is the average number of trees required to reach that performance. Overtraining never seems to be a problem for these weak learners, that is, as one increases the number of trees, the ensemble test error rate asymptotes and never increases.

Table 1. For fraction $f$ of difficult examples, the error rate for a single tree and a boosting ensemble and the number of trees required to reach the error rate for that ensemble.

| $f$ | single tree | boosting trees | number of trees |
|-----|-------------|----------------|-----------------|
| .1  | 12%         | 3.5%           | 25              |
| .3  | 13          | 4.5            | 28              |
| .5  | 16          | 7.1            | 31              |
| .7  | 21          | 7.7            | 60              |
| .9  | 23          | 8.1            | 72              |

We wanted to compare the boosting ensemble to other techniques for constructing ensembles using 14,000 examples, holding out 2000 for testing. The problem with decision trees is that invariably, even if the training data is different (but drawn from the same distribution), the features chosen for the first few nodes are usually the same (at least for the OCR data). Thus, different decision surfaces are not created. In order to create different decision regions for each tree, we can force each decision tree to consider another attribute as the root node, perhaps choosing that attribute from the first few attributes with largest information ratio. This is similar to what Kwok and Carter (1990) have suggested but we have many more trees and their interactive approach did not look feasible here. Another technique suggested by T.K. Ho (1992) is to construct independent trees on the same 10,000 examples but randomly striking out the use of fifty of the 100 possible features. Thus, for each tree, we randomly pick 50 features to construct the tree. When we use up to ten trees, the results using Ho's technique gives similar results to that of boosting but the asymptotic performance is far better for boosting. After we had performed these experiments, we learned of a technique termed "bagging" (Breiman, 1994) and we have yet to resolve the issue of whether bagging or boosting is better.

## 5 CONCLUSIONS

Based on preliminary evidence, it appears that for these applications a new boosting algorithm using trees as weak learners gives far superior performance to single trees and any other technique for constructing ensemble of trees. For boosting to work on any problem, one must find a weak learner that gives an error rate of less than 0.5 on the filtered training set. An important aspect of the building process is to prune based on a separate pruning set rather than pruning based on a training set. We have also tried this technique on knowledge discovery and data mining problems and the results are better than single neural networks.

## References

L. Bottou, C. Cortes, J.S. Denker, H. Drucker, I. Guyon, L.D. Jackel, Y. LeCun, U.A. Muller, E. Sackinger, P. Simard, and V. Vapnik (1994), "Comparison of Classifier Methods: A Case Study in Handwritten Digit Recognition", 1994 International Conference on Pattern Recognition, Jerusalem.

L. Breiman, J. Friedman, R.A. Olshen, and C.J. Stone (1984), *Classification and Regression Trees*, Chapman and Hall.

L. Breiman, "Bagging Predictors", Technical Report No. 421, Department of Statistics University of California, Berkeley, California 94720, September 1994.

H. Drucker (1994), C. Cortes, LD Jackel, Y. LeCun "Boosting and Other Ensemble Methods", *Neural Computation*, vol 6, no. 6, pp. 1287-1299.

H. Drucker, R.E. Schapire, and P. Simard (1993) "Boosting Performance in Neural Networks", *International Journal of Pattern Recognition and Artificial Intelligence*, Vol 7. No 4, pp. 705-719.

Y. Freund (1990), "Boosting a Weak Learning Algorithm by Majority", *Proceedings of the Third Workshop on Computational Learning Theory*, Morgan-Kaufmann, 202-216.

Y. Freund and R.E. Schapire (1995), "A decision-theoretic generalization of on-line learning and an application to boosting", *Proceeding of the Second European Conference on Computational Learning*.

T.K. Ho (1992), *A theory of Multiple Classifier Systems and Its Applications to Visual Word Recognition*, Doctoral Dissertation, Department of Computer Science, SUNY at Buffalo.

S.W. Kwok and C. Carter (1990), "Multiple Decision Trees", *Uncertainty in Artificial Intelligence 4*, R.D. Shachter, T.S. Levitt, L.N. Kanal, J.F Lemmer (eds) Elsevier Science Publishers.

J.R. Quinlan (1993), *C4.5: Programs For Machine Learning*, Morgan Kauffman.

J. Mingers (1989), "An Empirical Comparison of Pruning Methods for Decision Tree Induction", *Machine Learning*, 4:227-243.

R.E. Schapire (1990), The strength of weak learnability, *Machine Learning*, 5(2):197-227.
